# Temporal coding
# in the sub-millisecond range:
# Model of *barn owl* auditory pathway

**Richard Kempter***
Institut für Theoretische Physik
Physik-Department der TU München
D-85748 Garching bei München
Germany

**Wulfram Gerstner**
Institut für Theoretische Physik
Physik-Department der TU München
D-85748 Garching bei München
Germany

**J. Leo van Hemmen**
Institut für Theoretische Physik
Physik-Department der TU München
D-85748 Garching bei München
Germany

**Hermann Wagner**
Institut für Zoologie
Fakultät für Chemie und Biologie
D-85748 Garching bei München
Germany

## Abstract

Binaural coincidence detection is essential for the localization of external sounds and requires auditory signal processing with high temporal precision. We present an integrate-and-fire model of spike processing in the auditory pathway of the barn owl. It is shown that a temporal precision in the microsecond range can be achieved with neuronal time constants which are at least one magnitude longer. An important feature of our model is an unsupervised Hebbian learning rule which leads to a temporal fine tuning of the neuronal connections.

*email: kempter,wgerst,lvh @ physik.tu-muenchen.de

## 1  Introduction

Owls are able to locate acoustic signals based on extraction of interaural time difference by coincidence detection [1, 2]. The spatial resolution of sound localization found in experiments corresponds to a temporal resolution of auditory signal processing well below one millisecond. It follows that both the firing of spikes and their transmission along the so-called time pathway of the auditory system must occur with high temporal precision.

Each neuron in the nucleus magnocellularis, the second processing stage in the ascending auditory pathway, responds to signals in a narrow frequency range. Its spikes are phase locked to the external signal (Fig. 1a) for frequencies up to 8 kHz [3]. Axons from the nucleus magnocellularis project to the nucleus laminaris where signals from the right and left ear converge. Owls use the interaural phase difference for azimuthal sound localization. Since barn owls can locate signals with a precision of one degree of azimuthal angle, the temporal precision of spike encoding and transmission must be at least in the range of some 10 $\mu$s.

This poses at least two severe problems. First, the neural architecture has to be adapted to operating with high temporal precision. Considering the fact that the total delay from the ear to the nucleus magnocellularis is approximately 2-3 ms [4], a temporal precision of some 10 $\mu$s requires some fine tuning, possibly based on learning. Here we suggest that Hebbian learning is an appropriate mechanism. Second, neurons must operate with the necessary temporal precision. A firing precision of some 10 $\mu$s seems truly remarkable considering the fact that the membrane time constant is probably in the millisecond range. Nevertheless, it is shown below that neuronal spikes can be transmitted with the required temporal precision.

## 2  Neuron model

We concentrate on a single frequency channel of the auditory pathway and model a neuron of the nucleus magnocellularis. Since synapses are directly located on the *soma*, the spatial structure of the neuron can be reduced to a single compartment. In order to simplify the dynamics, we take an integrate-and-fire unit. Its membrane potential changes according to

$$\frac{d}{dt}u = -\frac{u}{\tau_0} + I(t) \tag{1}$$

where $I(t)$ is some input and $\tau_0$ is the membrane time constant. The neuron fires, if $u(t)$ crosses a threshold $\vartheta = 1$. This defines a firing time $t_0$. After firing $u$ is reset to an initial value $u_0 = 0$. Since auditory neurons are known to be fast, we assume a membrane time constant of 2 ms. Note that this is shorter than in other areas of the brain, but still a factor of 4 longer than the period of a 2 kHz sound signal.

The magnocellular neuron receives input from several presynaptic neurons $1 \leq k \leq K$. Each input spike at time $t_k^f$ generates a current pulse which decays exponentially with a fast time constant $\tau_r = 0.02$ ms. The magnitude of the current pulse depends on the coupling strength $J_k$. The total input is

$$I(t) = \sum_{k,f} J_k \exp(\frac{t - t_k^f}{\tau_r}) \theta(t - t_k^f) \tag{2}$$

where $\theta(x)$ is the unit step function and the sum runs over all input spikes.

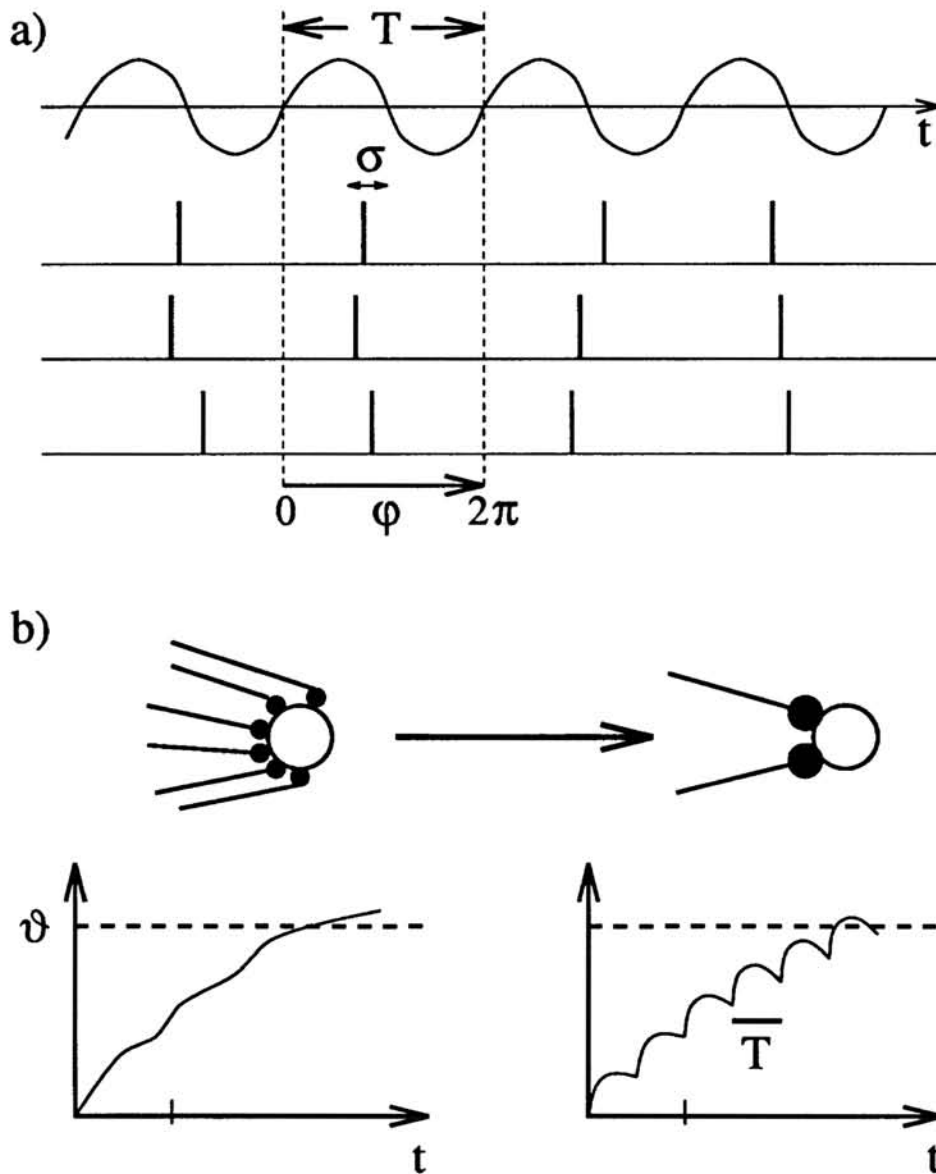

**Fig. 1.** *Principles of phase locking and learning.* a) The stimulus consists of a sound wave (top). Spikes of auditory nerve fibers leading to the nucleus magnocellularis are phase-locked to the periodic wave, that is, they occur at a preferred phase in relation to the sound, but with some jitter $\sigma$. Three examples of phase-locked spike trains are indicated. b) Before learning (left), many auditory input fibers converge to a neuron of the nucleus magnocellularis. Because of axonal delays which vary between different fibers, spikes *arrive* incoherently even though they are generated in a phase locked fashion. Due to averaging over several incoherent inputs, the total postsynaptic potential (bottom left) of a magnocellular neuron follows a rather smooth trajectory with no significant temporal structure. After learning (right) most connections have disappeared and only a few strong contacts remain. Input spikes now arrive coherently and the postsynaptic potential exhibits a clear oscillatory structure. Note that firing must occur during the rising phase of the oscillation. Thus output spikes will be phase locked.

All input signals belong to the same frequency channel with a carrier frequency of 2 kHz (period $T = 0.5$ ms), but the inputs arise from different presynaptic neurons ($1 \leq k \leq K$). Their axons have different diameter and length leading to a signal transmission delay $\Delta_k$ which varies between 2 and 3 ms [4]. Note that a delay as small as 0.25 ms shifts the signal by half a period.

Each input signal consists of a periodic spike train subject to two types of noise. First, a presynaptic neuron may not fire regularly every period but, on average, every $n^{th}$ period only where $n \approx 1/(\nu T)$ and $\nu$ is the mean firing rate of the neuron. For the sake of simplicity, we set $n = 1$. Second, the spikes may occur slightly too early or too late compared to the mean delay $\Delta$. Based on experimental results, we assume a typical shift $\sigma = \pm 0.05$ ms [3]. Specifically we assume in our model that inputs from a presynaptic neuron $k$ arrive with the probability density

$$P(t_k^f) = \frac{1}{\sqrt{2\pi}\sigma} \sum_{n=-\infty}^{\infty} \exp\left[ \frac{-(t_k^f - nT - \Delta_k)^2}{2\sigma^2} \right] \tag{3}$$

where $\Delta_k$ is the axonal transmission delay of input $k$ (Fig. 1).

## 3  Temporal tuning through learning

We assume a developmental period of unsupervised learning during which a fine tuning of the temporal characteristics of signal transmission takes place (Fig. 1b). Before learning the magnocellular neuron receives many inputs ($K = 50$) with weak coupling ($J_k = 1$). Due to the broad distribution of delays the total input (2) has, apart from fluctuations, no temporal structure. After learning, the magnocellular neuron receives input from two or three presynaptic neurons only. The connections to those neurons have become very effective; cf. Fig. 2.

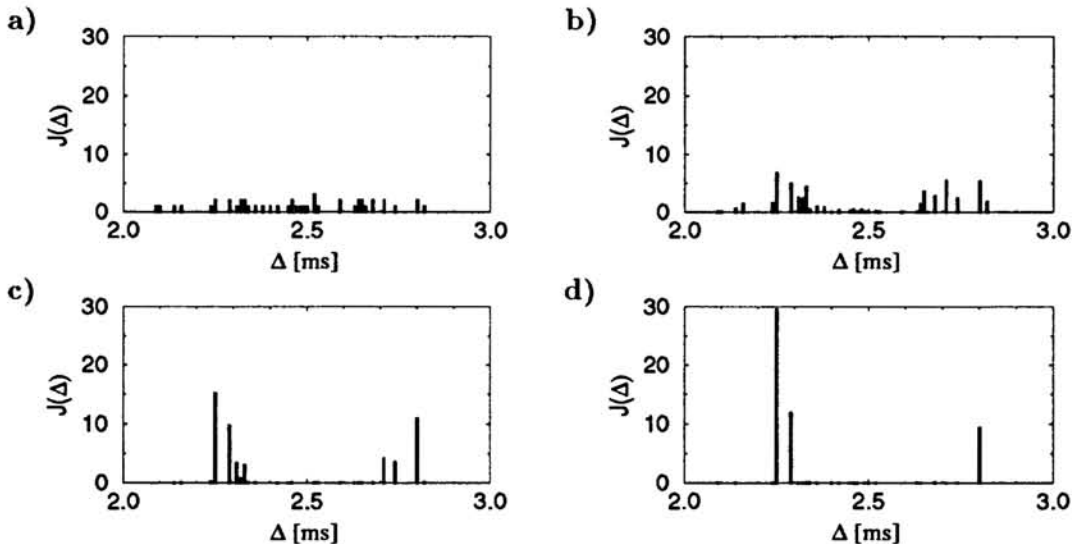

**Fig. 2.** *Learning. We plot the number of synaptic contacts (y-axis) for each delay $\Delta$ (x-axis). (a) At the beginning, the neuron has contacts to 50 presynaptic neurons with delays $2ms \leq \Delta \leq 3ms$. (b) and (c) During learning, some presynaptic neurons increase their number of contacts, other contacts disappear. (d) After learning, contacts to three presynaptic neurons with delays 2.25, 2.28, and 2.8 ms remain. The remaining contacts are very strong.*

The constant $J_k$ measures the *total* coupling strength between a presynaptic neuron $k$ and the postsynaptic neuron. Values of $J_k$ larger than one indicate that several synapses have been formed. It has been estimated from anatomical data that a fully developed magnocellular neuron receives inputs from as few as 1-4 presynaptic neurons, but each presynaptic axon shows multiple branching near the postsynaptic soma and makes up to one hundred synaptic contacts on the soma of the magnocellular neuron[5]. The result of our simulation study is consistent with this finding. In our model, learning leads to a final state with a few but highly effective inputs. The remaining inputs all have the same time delay modulo the period $T$ of the stimulus. Thus, learning leads to reduction of the number of input neurons contacts with a nucleus magnocellularis neuron. This is the fine tuning of the neuronal connections necessary for precise temporal coding (see below, section 4).

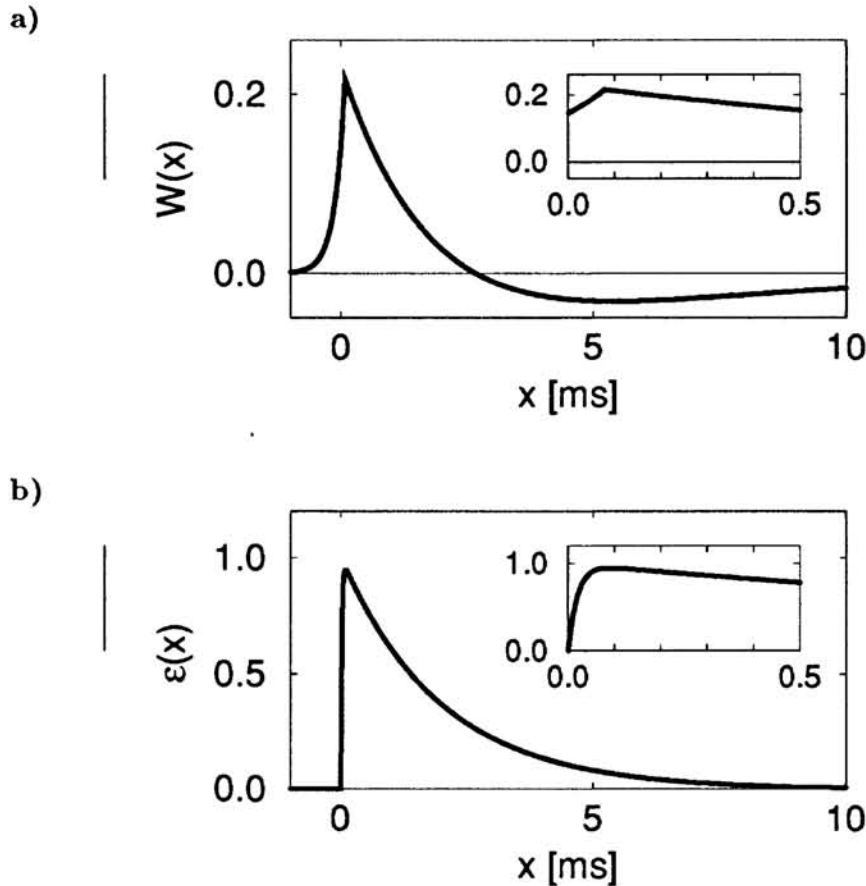

**Fig. 3.** *(a) Time window of learning $W(x)$. Along the x-axis we plot the time difference between presynaptic and postsynaptic fiing $x = t_i^f - t^k$. The window function $W(x)$ has a positive and a negative phase. Learning is most effective, if the postsynaptic spike is late by 0.08 ms (inset). (b) Postsynaptic potential $\epsilon(x)$. Each input spike evoked a postsynaptic potential which decays with a time constant of 2 ms. Since synapses are located directly at the soma, the rise time is very fast (see inset). Our learning scenario requires that the rise time of $\epsilon(x)$ should be approximately equal to the time $x$ where $W(x)$ has its maximum.*

In our model, temporal tuning is achieved by a variant of Hebbian learning. In standard Hebbian learning, synaptic weights are changed if pre- and postsynaptic activity occurs simultaneously. In the context of temporal coding by spikes, the concept of 'simultaneous activity' has to be refined. We assume that a synapse $k$ is

changed, if a presynaptic spike $t_k^f$ and a postsynaptic spike $t_0$ occur within a *time window* $W(t_k^f - t_0)$. More precisely, each pair of presynaptic and postsynaptic spikes changes a synapse $J_k$ by an amount

$$\Delta J_k = \gamma W(t_k^f - t_0) \tag{4}$$

with a prefactor $\gamma = 0.2$. Depending on the sign of $W(x)$, a contact to a presynaptic neuron is either increased or decreased. A decrease below $J_k = 0$ is not allowed. In our model, we assume a function $W(x)$ with two phases; cf. Fig. 3. For $x \approx 0$, the function $W(x)$ is positive. This leads to a strengthening (potentiation) of the contact with a presynaptic neuron $k$ which is active shortly before or after a postsynaptic spike. Synaptic contacts which become active more than 3 ms later than the postsynaptic spike are decreased. Note that the time window spans several cycles of length $T$. The combination of decrease and increase balances the average effects of potentiation and depression and leads to a normalization of the number and weight of synapses. Learning is stopped after 50.000 cycles of length $T$.

## 4  Temporal coding after learning

After learning contacts remain to a small number of presynaptic neurons. Their axonal transmission delays coincide or differ by multiples of the period $T$. Thus the spikes arriving from the few different presynaptic neurons have approximately the same phase and add up to an input signal (2) which retains, apart from fluctuations, the periodicity of the external sound signal (Fig.4a).

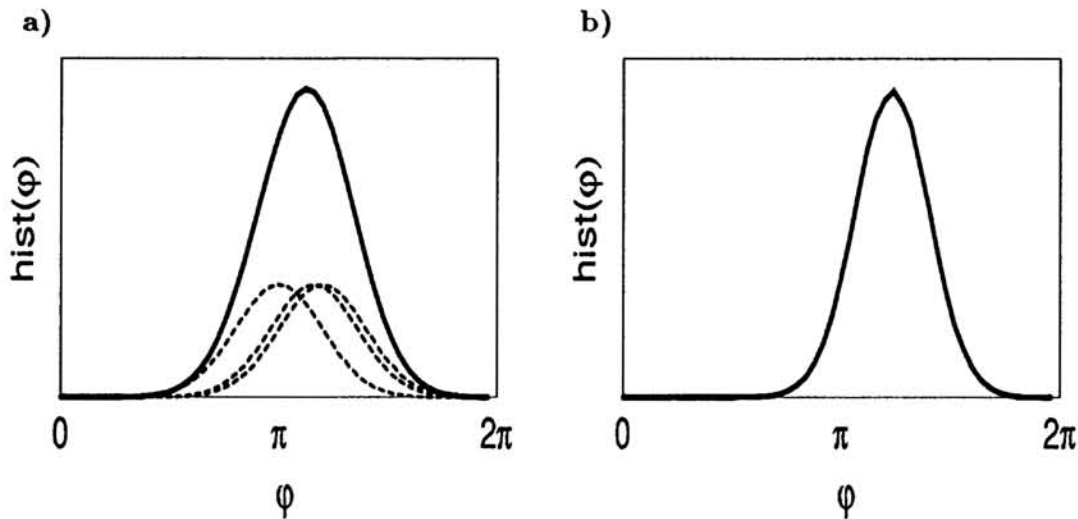

**Fig. 4.** *(a) Distribution of input phases after learning. The solid line shows the number of instances that an input spike with phase $\varphi$ has occured (arbitrary units). The input consists of spikes from the three presynaptic neurons which have survived after learning; cf. Fig. 1d. Due to the different delays, the mean input phase varies slightly between the three input channels. The dashed curves show the phase distribution of the individual channels, the solid line is the sum of the three dashed curves. (b) Distribution of output phases after learning. The histogram of output phases is sharply peaked. Comparison of the position of the maxima of the solid curves in (a) and (b) shows that the output is phase locked to the input with a relative delay $\delta\varphi$ which is related to the rise time of the postsynaptic potential.*

Output spikes of the magnocellular neuron are generated by the integrate-and-fire process (1). In Fig.4*b* we show a histogram of the phases of the output spikes. We find that the phases have a narrow distribution around a peak value. Thus the output is phase locked to the external signal. The width of the phase distribution corresponds to a precision of 0.084 phase cycles which equals 42 $\mu$s for a 2 kHz stimulus. Note that the temporal precision of the output has improved compared to the input where we had three channels with slightly different mean phases and a variation of $\sigma = 50\mu$s each. The increase in the precision is due to the average over three uncorrelated input signals.

We assume that the same principles are used during the following stages along the auditory pathway. In the nucleus laminaris several hundred signals are combined. This improves the signal-to-noise ratio further and a temporal precision below 10 $\mu$s could be achieved.

## 5 Discussion

We have demonstrated that precise temporal coding in the microsecond range is possible despite neuronal time constants in the millisecond range. Temporal refinement has been achieved through a slow developmental learning rule. It is a correlation based rule with a time window $W$ which spans several milliseconds. Nevertheless learning leads to a fine tuning of the connections supporting temporal coding with a resolution of 42 $\mu$s. The membrane time constant was set to 2 ms. This is nearly two orders of magnitudes longer than the achieved resolution. In our model, there is only one fast time constant which describes the typical duration of a input current pulse evoked by a presynaptic spike. Our value of $\tau_r = 20$ $\mu$s corresponds to a rise time of the postsynaptic potential of 100 $\mu$s. This seems to be realistic for auditory neurons since synaptic contacts are located directly on the *soma* of the postsynaptic neuron. The basic results of our model can also be applied to other areas of the brain and can shed new light on some aspects of temporal coding with slow neurons.

**Acknowledgments:** R.K. holds scholarship of the state of Bavaria. W.G. has been supported by the Deutsche Forschungsgemeinschaft (DFG) under grant number He 1729/2-2. H.W. is a Heisenberg fellow of the DFG.

## References

[1] L. A. Jeffress, J. Comp. Physiol. Psychol. **41**, 35 (1948).

[2] M. Konishi, Trends Neurosci. **9**, 163 (1986).

[3] C. E. Carr and M. Konishi, J. Neurosci. **10**, 3227 (1990).

[4] W. E. Sullivan and M. Konishi, J. Neurosci. **4**, 1787 (1984).

[5] C. E. Carr and R. E. Boudreau, J. Comp. Neurol. **314**, 306 (1991).
